# Predicting EMG Data from M1 Neurons with Variational Bayesian Least Squares

**Jo-Anne Ting**[1], **Aaron D'Souza**[1]
**Kenji Yamamoto**[3], **Toshinori Yoshioka**[2] , **Donna Hoffman**[3]
**Shinji Kakei**[4], **Lauren Sergio**[6], **John Kalaska**[5]
**Mitsuo Kawato**[2], **Peter Strick**[3], **Stefan Schaal**[1,2]

[1]Comp. Science & Neuroscience, U.of S. California, Los Angeles, CA 90089, USA
[2]ATR Computational Neuroscience Laboratories, Kyoto 619-0288, Japan
[3]University of Pittsburgh, Pittsburgh, PA 15261, USA
[4]Tokyo Metropolitan Institute for Neuroscience, Tokyo 183-8526, Japan
[5]University of Montreal, Montreal, Canada H3C-3J7
[6]York University, Toronto, Ontario, Canada M3J1P3

## Abstract

An increasing number of projects in neuroscience requires the statistical analysis of high dimensional data sets, as, for instance, in predicting behavior from neural firing or in operating artificial devices from brain recordings in brain-machine interfaces. Linear analysis techniques remain prevalent in such cases, but classical linear regression approaches are often numerically too fragile in high dimensions. In this paper, we address the question of whether EMG data collected from arm movements of monkeys can be faithfully reconstructed with linear approaches from neural activity in primary motor cortex (M1). To achieve robust data analysis, we develop a full Bayesian approach to linear regression that automatically detects and excludes irrelevant features in the data, regularizing against overfitting. In comparison with ordinary least squares, stepwise regression, partial least squares, LASSO regression and a brute force combinatorial search for the most predictive input features in the data, we demonstrate that the new Bayesian method offers a superior mixture of characteristics in terms of regularization against overfitting, computational efficiency and ease of use, demonstrating its potential as a drop-in replacement for other linear regression techniques. As neuroscientific results, our analyses demonstrate that EMG data can be well predicted from M1 neurons, further opening the path for possible real-time interfaces between brains and machines.

## 1 Introduction

In recent years, there has been growing interest in large scale analyses of brain activity with respect to associated behavioral variables. For instance, projects can be found in the area of brain-machine interfaces, where neural firing is directly used to control an artificial system like a robot [1, 2], to control a cursor on a computer screen via non-invasive brain signals [3] or to classify visual stimuli presented to

a subject [4, 5]. In these projects, the brain signals to be processed are typically high dimensional, on the order of hundreds or thousands of inputs, with large numbers of redundant and irrelevant signals. Linear modeling techniques like linear regression are among the primary analysis tools [6, 7] for such data. However, the computational problem of data analysis involves not only data fitting, but requires that the model extracted from the data has good generalization properties. This is crucial for predicting behavior from future neural recordings, e.g., for continual online interpretation of brain activity to control prosthetic devices or for longitudinal scientific studies of information processing in the brain. Surprisingly, robust linear modeling of high dimensional data is non-trivial as the danger of fitting noise and encountering numerical problems is high. Classical techniques like ridge regression, stepwise regression or partial least squares regression are known to be prone to overfitting and require careful human supervision to ensure useful results.

In this paper, we will focus on how to improve linear data analysis for the high dimensional scenarios described above, with a view towards developing a statistically robust "black box" approach that automatically detects the most relevant input dimensions for generalization and excludes other dimensions in a statistically sound way. For this purpose, we investigate a full Bayesian treatment of linear regression with automatic relevance detection [8]. Such an algorithm, called Variational Bayesian Least Squares (VBLS), can be formulated in closed form with the help of a variational Bayesian approximation and turns out to be computationally highly efficient. We apply VBLS to the reconstruction of EMG data from motor cortical firing, using data sets collected by [9] and [10, 11]. This data analysis addresses important neuroscientific questions in terms of whether M1 neurons can directly predict EMG traces [12], whether M1 has a muscle-based topological organization and whether information in M1 should be used to predict behavior in future brain-machine interfaces. Our main focus in this paper, however, will be on the robust statistical analysis of these kinds of data. Comparisons with classical linear analysis techniques and a brute force combinatorial model search on a cluster computer demonstrate that our VBLS algorithm achieves the "black box" quality of a robust statistical analysis technique without any tunable parameters.

In the following sections, we will first sketch the derivation of Variational Bayesian Least Squares and subsequently perform extensive comparative data analysis of this technique in the context of prediction EMG data from M1 neural firing.

## 2    High Dimensional Regression

Before developing our VBLS algorithm, let us briefly revisit classical linear regression techniques. The standard model for linear regression is:

$$y = \sum_{m=1}^{d} b_m x_m + \epsilon \tag{1}$$

where $\mathbf{b}$ is the regression vector composed of $b_m$ components, $d$ is the number of input dimensions, $\epsilon$ is additive mean-zero noise, $\mathbf{x}$ are the inputs and $y$ are the outputs. The Ordinary Least Squares (OLS) estimate of the regression vector is $\mathbf{b} = \left(\mathbf{X}^T \mathbf{X}\right)^{-1} \mathbf{X}^T \mathbf{y}$. The main problem with OLS regression in high dimensional input spaces is that the full rank assumption of $\left(\mathbf{X}^T \mathbf{X}\right)^{-1}$ is often violated due to underconstrained data sets. Ridge regression can "fix" such problems numerically, but introduces uncontrolled bias. Additionally, if the input dimensionality exceeds around 1000 dimensions, the matrix inversion can become prohibitively computationally expensive.

Several ideas exist how to improve over OLS. First, stepwise regression [13] can be employed. However, it has been strongly criticized for its potential for overfitting and its inconsistency in the presence of collinearity in the input data [14]. To

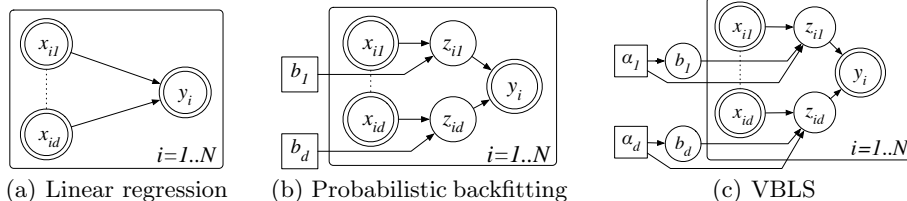

| (a) Linear regression | (b) Probabilistic backfitting | (c) VBLS |

Figure 1: Graphical Models for Linear Regression. Random variables are in circular nodes, observed random variables are in double circles and point estimated parameters are in square nodes.

deal with such collinearity directly, dimensionality reduction techniques like Principal Components Regression (PCR) and Factor Regression (FR) [15] are useful. These methods retain components in input space with large variance, regardless of whether these components influence the prediction [16], and can even eliminate low variance inputs that may have high predictive power for the outputs [17]. Another class of linear regression methods are projection regression techniques, most notably Partial Least Squares Regression (PLS) [18]. PLS performs computationally inexpensive $O(d)$ univariate regressions along projection directions, chosen according to the correlation between inputs and outputs. While slightly heuristic in nature, PLS is a surprisingly successful algorithm for ill-conditioned and high-dimensional regression problems, although it also has a tendency towards overfitting [16]. LASSO (Least Absolute Shrinkage and Selection Operator) regression [19] shrinks certain regression coefficients to 0, giving interpretable models that are sparse. However, a tuning parameter needs to be set, which can be done using n-fold cross-validation or manual hand-tuning. Finally, there are also more efficient methods for matrix inversion [20, 21], which, however, assume a well-condition regression problem a priori and degrade in the presence of collinearities in inputs.

In the following section, we develop a linear regression algorithm in a Bayesian framework that automatically regularizes against problems of overfitting. Moreover, the iterative nature of the algorithm, due to its formulation as an Expectation-Maximization problem [22], avoids the computational cost and numerical problems of matrix inversions. Thus, it addresses the two major problems of high-dimensional OLS simultaneously. Conceptually, the algorithm can be interpreted as a Bayesian version of either backfitting or partial least squares regression.

## 3   Variational Bayesian Least Squares

Figure 1 illustrates the progression of graphical models that we need in order to develop a robust Bayesian version of linear regression. Figure 1a depicts the standard linear regression model. In the spirit of PLS, if we knew an optimal projection direction of the input data, then the entire regression problem could be solved by a univariate regression between the projected data and the outputs. This optimal projection direction is simply the true gradient between inputs and outputs. In the tradition of EM algorithms [22], we encode this projection direction as a hidden variable, as shown in Figure 1b. The unobservable variables $z_{im}$ (where $i = 1..N$ denotes the index into the data set of $N$ data points) are the results of each input being multiplied with its corresponding component of the projection vector (i.e. $b_m$). Then, the $z_{im}$ are summed up to form a predicted output $y_i$.

More formally, the linear regression model in Eq. (1) is modified to become:

$$z_{im} = b_m x_{im} \qquad\qquad y_i = \sum_{m=1}^{d} z_{im} + \epsilon$$

For a probabilistic treatment with EM, we make a standard normal assumption of all distributions in form of:

$$y_i | \mathbf{z}_i \sim \text{Normal}\left(y_i; \mathbf{1}^T \mathbf{z}_i, \psi_y\right) \qquad z_{im} | \mathbf{x}_i \sim \text{Normal}\left(z_{im}; b_m x_{im}, \psi_{zm}\right)$$

where $\mathbf{1} = [1, 1, .., 1]^T$. While this model is still identical to OLS, notice that in the graphical model, the regression coefficients $b_m$ are behind the fan-in to the outputs $y_i$. Given the data $D = \{\mathbf{x}_i, y_i\}_{i=1}^N$, we can view this new regression model as an EM problem and maximize the incomplete log likelihood $\log p(\mathbf{y}|\mathbf{X})$ by maximizing the expected complete log likelihood $\langle \log p(\mathbf{y}, \mathbf{Z}|\mathbf{X}) \rangle$:

$$\log p(\mathbf{y}, \mathbf{Z}|\mathbf{X}) = -\frac{N}{2} \log \psi_y - \frac{1}{2\psi_y} \sum_{i=1}^N \left(y_i - \mathbf{1}^T \mathbf{z}_i\right)^2 - \frac{N}{2} \sum_{m=1}^d \log \psi_{zm}$$
$$- \sum_{m=1}^d \frac{1}{2\psi_{zm}} \left(z_{im} - b_m x_{im}\right)^2 + \text{const} \tag{2}$$

where $\mathbf{Z}$ denotes the $N$ by $d$ matrix of all $z_{im}$. The resulting EM updates require standard manipulations of normal distributions and result in:

**M-step** :                               **E-step** :

$$b_m = \frac{\sum_{i=1}^N \langle z_{im} \rangle x_{im}}{\sum_{i=1}^N x_{im}^2} \qquad\qquad \mathbf{1}^T \boldsymbol{\Sigma}_z \mathbf{1} = \left(\sum_{m=1}^d \psi_{zm}\right) \left[1 - \frac{1}{s}\left(\sum_{m=1}^d \psi_{zm}\right)\right]$$

$$\psi_y = \frac{1}{N} \sum_{i=1}^N \left(y_i - \mathbf{1}^T \langle \mathbf{z}_i \rangle\right) 2 + \mathbf{1}^T \boldsymbol{\Sigma}_z \mathbf{1} \qquad \sigma_{zm}^2 = \psi_{zm}\left(1 - \frac{1}{s}\psi_{zm}\right)$$

$$\psi_{zm} = \frac{1}{N} \sum_{i=1}^N \left(\langle z_{im} \rangle - b_m x_{im}\right)^2 + \sigma_{zm}^2 \qquad \langle z_{im} \rangle = b_m \mathbf{x}_i + \frac{1}{s}\psi_{xm}\left(y_i - \mathbf{b}^T \mathbf{x}_i\right)$$

where we define $s = \psi_y + \sum_{m=1}^d \psi_{xm}$ and $\boldsymbol{\Sigma}_z = \text{Cov}(\mathbf{z}|\mathbf{y}, \mathbf{X})$. It is very important to note that one EM update has a computational complexity of $O(d)$, where $d$ is the number of input dimensions, instead of the $O(d^3)$ associated with OLS regression. This efficiency comes at the cost of an iterative solution, instead of a one-shot solution for $\mathbf{b}$ as in OLS. It can be proved that this EM version of least squares regression is guaranteed to converge to the same solution as OLS [23].

This new EM algorithm appears to only replace the matrix inversion in OLS by an iterative method, as others have done with alternative algorithms [20, 21], although the convergence guarantees of EM are an improvement over previous approaches. The true power of this probabilistic formulation, though, becomes apparent when we add a Bayesian layer that achieves the desired robustness in face of ill-conditioned data.

## 3.1    Automatic Relevance Determination

From a Bayesian point of view, the parameters $b_m$ should be treated probabilistically so that we can integrate them out to safeguard against overfitting. For this purpose, as shown in Figure 1c, we introduce precision variables $\alpha_m$ over each regression parameter $b_m$:

$$p(\mathbf{b}|\boldsymbol{\alpha}) = \prod_{m=1}^d \left(\frac{\alpha_m}{2\pi}\right)^{\frac{1}{2}} \exp\left\{-\frac{\alpha_m}{2} b_m^2\right\}$$
$$p(\boldsymbol{\alpha}) = \prod_{m=1}^d \frac{b_\alpha^{a_\alpha}}{\text{Gamma}(a_\alpha)} \alpha_m^{(a_\alpha - 1)} \exp\left\{-b_\alpha \alpha_m\right\} \tag{3}$$

where $\boldsymbol{\alpha}$ is the vector of all $\alpha_m$. In order to obtain a tractable posterior distribution over all hidden variables $\mathbf{b}$, $z_{im}$ and $\boldsymbol{\alpha}$, we use a factorial variational approximation of the true posterior $Q(\boldsymbol{\alpha}, \mathbf{b}, \mathbf{Z}) = Q(\boldsymbol{\alpha}, \mathbf{b})Q(\mathbf{Z})$. Note that the connection from the $\alpha_m$ to the corresponding $z_{im}$ in Figure 1c is an intentional design. Under this graphical model, the marginal distribution of $b_m$ becomes a Student $t$-distribution that allows traditional hypothesis testing [24]. The minimal factorization of the posterior into $Q(\boldsymbol{\alpha}, \mathbf{b})Q(\mathbf{Z})$ would not be possible without this special design.

The resulting augmented model has the following distributions:

$$y_i | \mathbf{z}_i \sim N(y_i; \mathbf{1}^T \mathbf{z}_i, \psi_y) \qquad\qquad b_m | \alpha_m \sim N(w_{bm}; 0, 1/\alpha_m)$$
$$z_{im} | b_m, \alpha_m x_{im} \sim N(z_{im}; b_m x_{im}, \psi_{zm}/\alpha_m) \qquad \alpha_m \sim \text{Gamma}(\alpha_m; a_\alpha, b_\alpha)$$

We now have a mechanism that infers the significance of each dimension's contribution to the observed output $y$. Since $b_m$ is zero mean, a very large $\alpha_m$ (equivalent to a very small variance of $b_m$) suggests that $b_m$ is very close to 0 and has no contribution to the output. An EM-like algorithm [25] can be used to find the posterior updates of all distributions. We omit the EM update equations due to space constraints as they are similar to the EM update above and only focus on the posterior update for $b_m$ and $\alpha$:

$$\sigma^2_{b_m|\alpha_m} = \frac{\psi_{zm}}{\alpha_m} \left( \sum_{i=1}^N x_{im}^2 + \psi_{zm} \right)^{-1}$$

$$\langle b_m|\alpha_m \rangle = \left( \sum_{i=1}^N x_{im}^2 + \psi_{zm} \right)^{-1} \left( \sum_{i=1}^N \langle z_{im} \rangle x_{im} \right)$$

$$\hat{a}_\alpha = a_\alpha + \frac{N}{2} \tag{4}$$

$$\hat{b}_\alpha^{(m)} = b_\alpha + \frac{1}{2\psi_{zm}} \left\{ \sum_{i=1}^N \langle z_{im}^2 \rangle - \left( \sum_{i=1}^N x_{im}^2 + \psi_{zm} \right)^{-1} \left( \sum_{i=1}^N \langle z_{im} \rangle x_{im} \right)^2 \right\}$$

Note that the update equation for $\langle b_m|\alpha_m \rangle$ can be rewritten as:

$$\langle b_m|\alpha_m \rangle^{(n+1)} = \left( \frac{\sum_{i=1}^N x_{im}^2}{\sum_{i=1}^N x_{im}^2 + \psi_{zm}} \right) \langle b_m|\alpha_m \rangle^{(n)} + \frac{\psi_{zm}}{s\alpha_m} \frac{\sum_{i=1}^N \left( y_i - \langle \mathbf{b}|\boldsymbol{\alpha} \rangle^{(n)T} \mathbf{x}_i \right) x_{im}}{\sum_{i=1}^N x_{im}^2 + \psi_{zm}} \tag{5}$$

Eq. (5) demonstrates that in the absence of a correlation between the current input dimension and the residual error, the first term causes the current regression coefficient to decay. The resulting regression solution regularizes over the number of retained inputs in the final regression vector, performing a functionality similar to Automatic Relevance Determination (ARD) [8]. The update equations' algorithmic complexity remains $O(d)$. One can further show that the marginal distribution of all $b_m$ is a $t$-distribution with $t = \langle b_m|\alpha_m \rangle / \sigma_{b_m|\alpha_m}$ and $2\hat{a}_\alpha$ degrees of freedom, which allows a principled way of determining whether a regression coefficient was excluded by means of standard hypothesis testing. Thus, Variational Bayesian Least Squares (VBLS) regression is a full Bayesian treatment of the linear regression problem.

## 4  Evaluation

We now turn to the application and evaluation of VBLS in the context of predicting EMG data from neural data recorded in M1 of monkeys. The key questions addressed in this application were i) whether EMG data can be reconstructed accurately with good generalization, ii) how many neurons contribute to the reconstruction of each muscle and iii) how well the VBLS algorithm compares to other analysis techniques. The underlying assumption of this analysis is that the relationship between neural firing and muscle activity is approximately linear.

### 4.1  Data sets

We investigated data from two different experiments. In the first experiment by Sergio & Kalaska [9], the monkey moved a manipulandum in a center-out task in eight different directions, equally spaced in a horizontal planar circle of 8cm radius. A variation of this experiment held the manipulandum rigidly in place, while the monkey applied isometric forces in the same eight directions. In both conditions, movement or force, feedback was given through visual display on a monitor. Neural activity for 71 M1 neurons was recorded in all conditions (2400 data points for each neuron), along with the EMG outputs of 11 muscles.

The second experiment by Kakei et al. [10] involved a monkey trained to perform eight different combinations of wrist flexion-extension and radial-ulnar movements while in three different arm postures (pronated, supinated and midway between the two). The data set consisted of neural data of 92 M1 neurons that were recorded

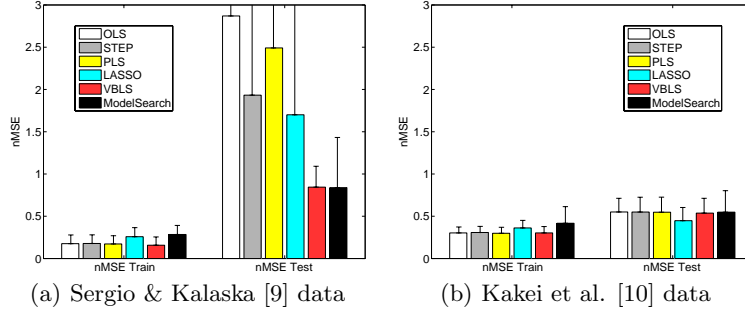

(a) Sergio & Kalaska [9] data     (b) Kakei et al. [10] data

Figure 2: Normalized mean squared error for Cross-validation Sets (6-fold for [10] and 8-fold for [9])

|  | VBLS | PLS | STEP | LASSO |
|---|---|---|---|---|
| Sergio & Kalaska data set | 93.6% | 7.44% | 8.71% | 8.42% |
| Kakei et al. data set | 87.1% | 40.1% | 72.3% | 76.3% |

Table 1: Percentage neuron matches between baseline and all other algorithms, averaged over all muscles in the data set

at all three wrist postures (producing 2664 data points for each neuron) and the EMG outputs of 7 contributing muscles. In all experiments, the neural data was represented as average firing rates and was time aligned with EMG data based on analyses that are outside of the scope of this paper.

## 4.2  Methods

For the Sergio & Kalaska data set, a baseline comparison of good EMG reconstruction was obtained through a limited combinatorial search over possible regression models. A particular model is characterized by a subset of neurons that is used to predict the EMG data. Given 71 neurons, theoretically $2^{71}$ possible models exist. This value is too large for an exhaustive search. Therefore, we considered only possible combinations of up to 20 neurons, which required several weeks of computation on a 30-node cluster computer. The optimal predictive subset of neurons was determined from an 8-fold cross validation. This baseline study served as a comparison for PLS, stepwise regression, LASSO regression, OLS and VBLS. The five other algorithms used the same validation sets employed in the baseline study. The number of PLS projections for each data fit was found by leave-one-out cross-validation. Stepwise regression used Matlab's "stepwisefit" function. LASSO regression was implemented, manually choosing the optimal tuning parameter over all cross-validation sets. OLS was implemented using a small ridge regression parameter of $10^{-10}$ in order to avoid ill-conditioned matrix inversions.

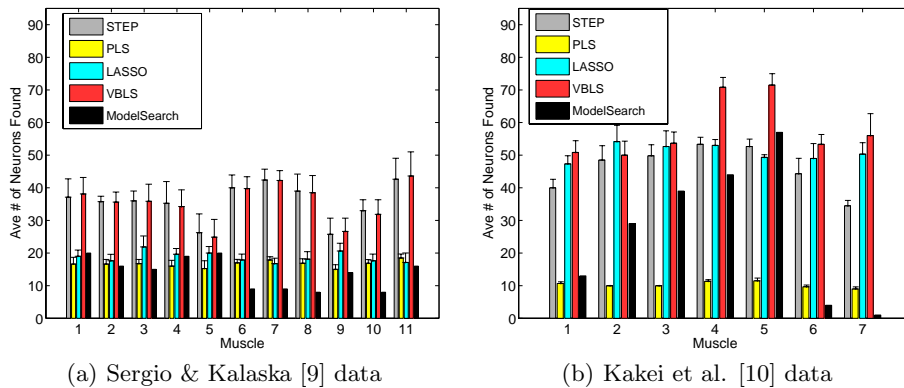

(a) Sergio & Kalaska [9] data     (b) Kakei et al. [10] data

Figure 3: Average Number of Relevant Neurons found over Cross-validation Sets (6-fold for [10] and 8-fold for [9])

The average number of relevant neurons was calculated over all 8 cross-validation sets and a final set of relevant neurons was reached for each algorithm by taking the common neurons found to be relevant over the 8 cross-validation sets. Inference of relevant neurons in PLS was based on the subspace spanned by the PLS projections, while relevant neurons in VBLS were inferred from $t$-tests on the regression parameters, using a significance of $p < 0.05$. Stepwise regression and LASSO regression determined the number of relevant neurons from the inputs that were included in the final model. Note that since OLS retained all input dimensions, this algorithm was omitted in relevant neuron comparisons.

Analogous to the first data set, a combinatorial analysis was performed on the Kakei et al. data set in order to determine the optimal set of neurons contributing to each muscle (i.e. producing the lowest possible prediction error) in a 6-fold cross-validation. PLS, stepwise regression, LASSO regression, OLS and VBLS were applied using the same cross-validation sets, employing the same procedure described for the first data set.

## 4.3 Results

Figure 2 shows that, in general, EMG traces seem to be well predictable from M1 neural firing. VBLS resulted in a generalization error comparable to that produced by the baseline study. In the Kakei et al. dataset, all algorithms performed similarly, with LASSO regression performing a little better than the rest. However, OLS, stepwise regression, LASSO regression and PLS performed far worse on the Sergio & Kalaska dataset, with OLS regression attaining the worst error. Such performance is typical for traditional linear regression methods on ill-conditioned high dimensional data, motivating the development of VBLS. The average number of relevant neurons found by VBLS was slightly higher than the baseline study, as seen in Figure 3. This result is not surprising as the baseline study did not consider all possible combination of neurons. Given the good generalization results of VBLS, it seems that the Bayesian approach regularized the participating neurons sufficiently so that no overfitting occurred. Note that the results for muscle 6 and 7 in Figure 3b seem to be due to some irregularities in the data and should be considered outliers. Table 1 demonstrates that the relevant neurons identified by VBLS coincided at a very high percentage with those of the baseline results, while PLS, stepwise regression and LASSO regression had inferior outcomes.

Thus, in general, VBLS achieved comparable performance with the baseline study when reconstructing EMG data from M1 neurons. While VBLS is an iterative statistical method, which performs slower than classical "one-shot" linear least squares methods (i.e., on the order of several minutes for the data sets in our analyses), it achieved comparable results with our combinatorial model search, which took weeks on a cluster computer.

## 5 Discussion

This paper addressed the problem of analyzing high dimensional data with linear regression techniques, as encountered in neuroscience and the new field of brain-machine interfaces. To achieve robust statistical results, we introduced a novel Bayesian technique for linear regression analysis with automatic feature detection, called Variational Bayesian Least Squares. Comparisons with classical linear regression methods and a "gold standard" obtained from a brute force search over all possible linear models demonstrate that VBLS performs very well without any manual parameter tuning, such that it has the quality of a "black box" statistical analysis technique.

A point of concern against the VBLS algorithm is how the variational approximation in this algorithm affects the quality of function approximation. It is known that factorial approximations to a joint distribution create more peaked distributions, such that one could potentially assume that VBLS might tend to overfit. However, in the case of VBLS, a more peaked distribution over $b_m$ pushes the regression parameter closer to zero. Thus, VBLS will be on the slightly pessimistic side of function fitting and is unlikely to overfit. Future evaluations and comparisons with Markov Chain Monte Carlo methods will reveal more details of the nature of the variational approximation. Regardless, it appears that VBLS could become a useful drop-in replacement for various classical regression methods. It lends itself to incremental implementation as would be needed in real-time analyses of brain information.

## Acknowledgments

This research was supported in part by National Science Foundation grants ECS-0325383, IIS-0312802, IIS-0082995, ECS-0326095, ANI-0224419, a NASA grant AC#98 − 516, an AFOSR grant on Intelligent Control, the ERATO Kawato Dynamic Brain Project funded by the Japanese Science and Technology Agency, the ATR Computational Neuroscience Laboratories and by funds from the Veterans Administration Medical Research Service.

# References

[1] M.A. Nicolelis. Actions from thoughts. *Nature*, 409:403–407, 2001.

[2] D.M. Taylor, S.I. Tillery, and A.B. Schwartz. Direct cortical control of 3d neuroprosthetic devices. *Science*, 296:1829–1932, 2002.

[3] J.R. Wolpaw and D.J. McFarland. Control of a two-dimensional movement signal by a noninvasive brain-computer interface in humans. *Proceedings of the National Academy of Sciences*, 101:17849–17854, 2004.

[4] Y. Kamitani and F. Tong. Decoding the visual and subjective contents of the human brain. *Nature Neuroscience*, 8:679, 2004.

[5] J.D. Haynes and G. Rees. Predicting the orientation of invisible stimuli from activity in human primary visual cortex. *Nature Neuroscience*, 8:686, 2005.

[6] J. Wessberg and M.A. Nicolelis. Optimizing a linear algorithm for real-time robotic control using chronic cortical ensemble recordings in monkeys. *Journal of Cognitive Neuroscience*, 16:1022–1035, 2004.

[7] S. Musallam, B.D. Corneil, B. Greger, H. Scherberger, and R.A. Andersen. Cognitive control signals for neural prosthetics. *Science*, 305:258–262, 2004.

[8] R.M. Neal. *Bayesian learning for neural networks*. PhD thesis, Dept. of Computer Science, University of Toronto, 1994.

[9] L.E. Sergio and J.F. Kalaska. Changes in the temporal pattern of primary motor cortex activity in a directional isometric force versus limb movement task. *Journal of Neurophysiology*, 80:1577–1583, 1998.

[10] S. Kakei, D.S. Hoffman, and P.L. Strick. Muscle and movement representations in the primary motor cortex. *Science*, 285:2136–2139, 1999.

[11] S. Kakei, D.S. Hoffman, and P.L. Strick. Direction of action is represented in the ventral premotor cortex. *Nature Neuroscience*, 4:1020–1025, 2001.

[12] E. Todorov. Direct cortical control of muscle activation in voluntary arm movements: a model. *Nature Neuroscience*, 3:391–398, 2000.

[13] N. R. Draper and H. Smith. *Applied Regression Analysis*. Wiley, 1981.

[14] S. Derksen and H.J. Keselman. Backward, forward and stepwise automated subset selection algorithms: Frequency of obtaining authentic and noise variables. *British Journal of Mathematical and Statistical Psychology*, 45:265–282, 1992.

[15] W.F. Massey. Principal component regression in exploratory statistical research. *Journal of the American Statistical Association*, 60:234–246, 1965.

[16] S. Schaal, S. Vijayakumar, and C.G. Atkeson. Local dimensionality reduction. In M.I. Jordan, M.J. Kearns, and S.A. Solla, editors, *Advances in Neural Information Processing Systems*. MIT Press, 1998.

[17] I.E. Frank and J.H. Friedman. A statistical view of some chemometric regression tools. *Technometrics*, 35:109–135, 1993.

[18] H. Wold. Soft modeling by latent variables: The nonlinear iterative partial least squares approach. In J. Gani, editor, *Perspectives in probability and statistics, papers in honor of M. S. Bartlett*. Academic Press, 1975.

[19] R. Tibshirani. Regression shrinkage and selection via the lasso. *Journal of Royal Statistical Society, Series B*, 58(1):267–288, 1996.

[20] V. Strassen. Gaussian elimination is not optimal. *Num Mathematik*, 13:354–356, 1969.

[21] T. J. Hastie and R. J. Tibshirani. *Generalized additive models*. Number 43 in Monographs on Statistics and Applied Probability. Chapman and Hall, 1990.

[22] A. Dempster, N. Laird, and D. Rubin. Maximum likelihood from incomplete data via the em algorithm. *Journal of Royal Statistical Society. Series B*, 39(1):1–38, 1977.

[23] A. D'Souza, S. Vijayakumar, and S. Schaal. The bayesian backfitting relevance vector machine. In *Proceedings of the 21st International Conference on Machine Learning*. ACM Press, 2004.

[24] A. Gelman, J. Carlin, H.S. Stern, and D.B. Rubin. *Bayesian Data Analaysis*. Chapman and Hall, 2000.

[25] Z. Ghahramani and M.J. Beal. Graphical models and variational methods. In D. Saad and M. Opper, editors, *Advanced Mean Field Methods - Theory and Practice*. MIT Press, 2000.
